# Developing Topography and Ocular Dominance Using two aVLSI Vision Sensors and a Neurotrophic Model of Plasticity

**Terry Elliott**
Dept. Electronics & Computer Science
University of Southampton
Highfield
Southampton, SO17 1BJ
United Kingdom
*te@ecs.soton.ac.uk*

**Jörg Kramer**
Institute of Neuroinformatics
University of Zürich and ETH Zürich
Winterthurerstrasse 190
8057 Zürich
Switzerland
*kramer@ini.phys.ethz.ch*

## Abstract

A neurotrophic model for the co-development of topography and ocular dominance columns in the primary visual cortex has recently been proposed. In the present work, we test this model by driving it with the output of a pair of neuronal vision sensors stimulated by disparate moving patterns. We show that the temporal correlations in the spike trains generated by the two sensors elicit the development of refined topography and ocular dominance columns, even in the presence of significant amounts of spontaneous activity and fixed-pattern noise in the sensors.

## 1 Introduction

A large body of evidence suggests that the development of the retinogeniculocortical pathway, which leads in higher vertebrates to the emergence of eye-specific laminae in the lateral geniculate nucleus (LGN), the formation of ocular dominance columns (ODCs) in the striate cortex and the establishment of retinotopic representations in both structures, is a competitive, activity-dependent process (see Ref. [1] for a review). Experimental findings indicate that at least in the case of ODC formation, this competition may be mediated by retrograde neurotrophic factors (NTFs) [2]. A computational model for synaptic plasticity based on this hypothesis has recently been proposed [1]. This model has successfully been applied to the development and refinement of retinotopic representations in the LGN and striate cortex, and to the formation of ODCs in the striate cortex due to competition between the eye-specific laminae of the LGN. In this model, the activity within the afferent cell sheets was simulated either as interocularly uncorrelated spontaneous retinal waves or, as a coarse model of visually evoked activity, as interocularly correlated Gaussian noise. Gaussian noise, however, is not a realistic model of evoked retinal activity, nor do the interocular correlations introduced adequately capture the correlations that arise due to the spatial disparity between the two retinas.

For this study, we tested the ability of the plasticity model to generate topographic refinement and ODCs in response to afferent activity provided by a pair of biologically-inspired

artificial vision sensors. These sensors capture some of the properties of biological retinas. They convert optical images into analog electrical signals and perform brightness adaptation and logarithmic contrast-encoding. Their output is encoded in asynchronous, binary spike trains, as provided by the retinal ganglion cells of biological retinas. Mismatch of processing elements and temporal noise are a natural by-product of biological retinas and such vision sensors alike. One goal of this work was to determine the robustness of the model towards such nonidealities. While the refinement of topography from the temporal correlations provided by one vision sensor in response to moving stimuli has already been explored [3], the present work focuses on the co-development of topography and ODCs in response to the correlations between the signals from two vision sensors stimulated by disparate moving bars. In particular, the dependence of ODC formation on disparity and noise is considered.

## 2   Vision Sensor

The vision sensor used in the experiments is a two-dimensional array of $16 \times 16$ pixels fabricated with standard CMOS technology, where each pixel performs a two-way rectified temporal high-pass filtering operation on the incoming visual signal in the focal plane [4, 5]. The sensor adapts to background illuminance and responds to local positive and negative illuminance transients at separately coded terminals. The transients are converted into a stream of asynchronous binary pulses, which are multiplexed onto a common, arbitrated address bus, where the address encodes the location of the sending pixel and the sign of the transient. In the absence of any activity on the communication bus for a few hundred milliseconds the bus address decays to zero. A block diagram of a reduced-resolution array of pixels with peripheral arbitration and communication circuitry is shown in Fig. 1. Handshaking with external data acquisition circuitry is provided via the request ($REQ$) and acknowledge ($ACK$) terminals.

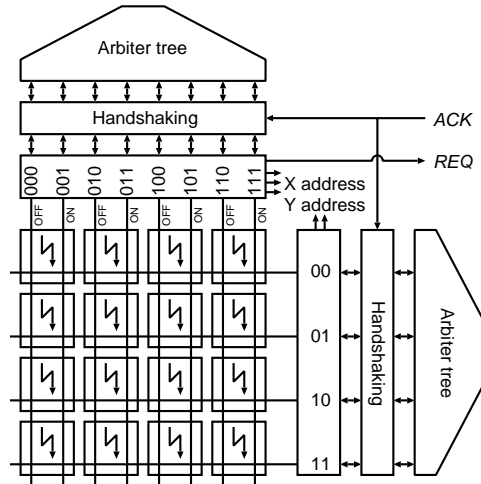

Figure 1: Block diagram of the sensor architecture (reduced resolution).

If the array is used for imaging purposes under constant or slowly-varying ambient lighting conditions, it only responds to boundaries or edges of moving objects or shadows of sufficient contrast and not to static scenes. Depending on the settings of different bias controls the imager can be used in different modes. Separate gain controls for ON and OFF transients permit the imager to respond to only one type of transient or to both types with adjustable weighting. Together with these gain controls, a threshold bias sets the contrast

response threshold and the rate of spontaneous activity. For sufficiently large thresholds, spontaneous activity is completely suppressed. Another bias control sets a refractory period that limits the maximum spike rate of each pixel. For short refractory periods, each contrast transient at a given pixel triggers a burst of spikes; for long refractory periods, a typical transient only triggers a single spike in the pixel, resulting in a very efficient, one-bit edge coding.

## 3 Sensor-Computer Interface

The two vision sensors were coupled to a computer via two parallel ports. The handshaking terminals of each chip were shorted, so that the sensors could operate at their own speed without being artificially slowed down by the computer. This avoided the risk of overloading the multiplexer and thereby distorting the data. Furthermore, this scheme was simpler to implement than a handshaking scheme. The lack of synchronization entailed several problems: missing out on events, reading events more than once, and reading spurious zero addresses in the absence of recent activity in the sensors. The first two problems could satisfactorily be solved by choosing a long refractory period, so that each moving-edge stimulus only evoked a single spike per pixel. For a typical stimulus this resulted in inter-spike intervals on the multiplexed bus of a few milliseconds, which made it unlikely that events would be missed. Furthermore, the refractory period prevented any given pixel from spiking more than once in a row in response to a moving edge, so that multiple reads of the same address were always due to the same event being read several times and therefore could be discarded. The ambiguity of the (0,0) address readings, namely whether such a reading meant that the (0,0) pixel was active or that the address on the bus had decayed to zero due to lack of activity, could not be resolved. It was therefore decided to ignore the (0,0) address and to exclude the (0,0) cell from each map. Using this strategy it was found that the data read by the computer reflected the optical stimuli with a small error rate.

## 4 Visual Stimulation

Two separate windows within the display of the LCD monitor of the computer used for data acquisition were each imaged onto one of the vision chips via a lens to provide the optical stimulation. The stimuli in each window consisted in eight separate sequences of images that were played without interruption, each new sequence being selected randomly after the completion of the previous one. Each sequence simulated a white bar sweeping across a black background. The sequences were distinguished only by the orientation and direction of motion of the bar, while the speed, as measured perpendicularly to the bar's orientation, was constant and identical for each sequence. The bar could have four different orientations, aligned to the rows or columns of the vision sensor or to one of the two diagonals, and move in either direction. The bars had a finite width of 20 pixels on the LCD display, corresponding to about 8 pixel periods on the image sensors, and they were sufficiently long entirely to fill the field of view of the chips. The displays in the two windows stimulating the two chips were identical save for a fixed relative displacement between the bars along the direction of motion during the entire run, simulating the disparity seen by two eyes looking at the same object. The used displacements were 0, 10, and 15 pixels on the LCD display, corresponding to no disparity and disparities of 1/2 the bar width (4 sensor pixels) and 3/4 of the bar width (6 sensor pixels), respectively. The speed of the bar was largely unimportant, because the output spikes of the chip were sampled into bins of fixed sizes, rather than bins representing fixed time windows. The chosen white bar on a black background stimulated the vision sensor with a leading ON edge and a trailing OFF edge. However, because the spurious activity of the chip, mainly in the form of crosstalk, was increased if both ON and OFF responses were activated and because we required only the response to one edge type for this work, the ON responses from the chip were suppressed.

# 5 Neurotrophic Model of Plasticity

Let the letters $i$ and $j$ label afferent cells within an afferent sheet, letters $\alpha$ and $\beta$ label the afferent sheets, and letters $x$ and $y$ label target cells. The two afferent sheets represent the two chips' arrays of pixels and are therefore $16\times16$ square arrays of cells. For convenience, the target array is also a $16\times16$ square array of cells. Let $a_i^\alpha$ denote an afferent cell's activity. For each time step of simulated development, we capture a fixed number of spikes from each chip. A pixel that has not spiked gives $a_i^\alpha = 0$, while one that has gives $a_i^\alpha = 1$. If $s_{xi}^\alpha$ represents the number of synapses projected from cell $i$ in afferent sheet $\alpha$ to target $x$, then $s_{xi}^\alpha$ evolves according to the equation

$$\frac{ds_{xi}^\alpha}{dt} = \epsilon s_{xi}^\alpha \left[ \frac{(a + a_i^\alpha)\rho_i^\alpha}{\sum_{\beta j} s_{xj}^\beta (a + a_j^\beta)\rho_j^\beta} \sum_y \Delta_{xy} \left( T_0 + T_1 \frac{\sum_{\beta j} s_{yj}^\beta a_j^\beta}{\sum_{\beta j} s_{yj}^\beta} \right) - 1 \right]. \quad (1)$$

Here, $T_0$ and $T_1$ represent, respectively, an activity-independent and a maximum activity-dependent release of NTF from target cells; the parameter $a$ a resting NTF uptake capacity by afferent cells; $\Delta_{xy}$ a function characterising NTF diffusion between target cells, which we take for convenience to be a Gaussian of width $\sigma$. The function $\rho_i^\alpha = \bar{a}_i^\alpha / \sum_x s_{xi}^\alpha$ is a simple model for the number of NTF receptors supported by an afferent cell, where $\bar{a}_i^\alpha$ denotes average afferent activity. The parameter $\epsilon$ sets the overall rate of development. Consistent with previous work [3], we set $\epsilon = 0.02$, $\sigma = 0.75$, $T_0 = 0$, $T_1 = 20$ and $a = 1$. Although this model appears complex, it can be shown to be equivalent to a non-linear Hebbian rule with competition implemented via multiplicative synaptic normalisation [6]. For a full discussion, derivation and justification of the model, see Ref. [7].

Both afferent sheets initially project roughly equally to all cells in the target sheet. The initial pattern of connectivity between the sheets is established following Goodhill's method [8]. For a given afferent cell, let $d$ be the distance between some target cell and the target cell to which the afferent cell would project were topography perfect; let $d_{\max}$ be the maximum such distance. Then the number of synapses projected by the afferent cell to this target cell is initially set to be proportional to

$$\beta \left( 1 - \frac{d}{d_{\max}} \right) + (1 - \beta)n, \quad (2)$$

where $n \in [0, 1]$ is a randomly selected number for each such pair of afferent and target cells. The parameter $\beta \in [0, 1]$ determines the quality of the projections, with $\beta = 1$ giving initially greatest topographical bias, so that an afferent cell projects maximally to its topographically preferred target cell, and $\beta = 0$ giving initially completely random projections. Here we set $\beta = 0.5$; the impact of decreasing $\beta$ on the final structure of the topographic map has been thoroughly explored elsewhere [3].

The topographic representation of an afferent sheet on the target sheet is depicted using standard methods [1, 8]: the centres of mass of afferent projections to all target cells are calculated, and these are then connected by lines that preserve the neighbourhood relations among the target cells.

# 6 Results

For each iteration step of the algorithm a fixed number of spikes was captured. The bin size determines the correlation space constants of the afferent cell sheets and therefore influences the final quality of the topographic mapping [3]. Unless otherwise noted the bin size was 32 per sensor, which corresponds to about two successive pixel rows stimulated by a moving contrast boundary. The presented simulations were performed for 15,000 to 20,000 iteration steps, sufficient for map development to be largely complete.

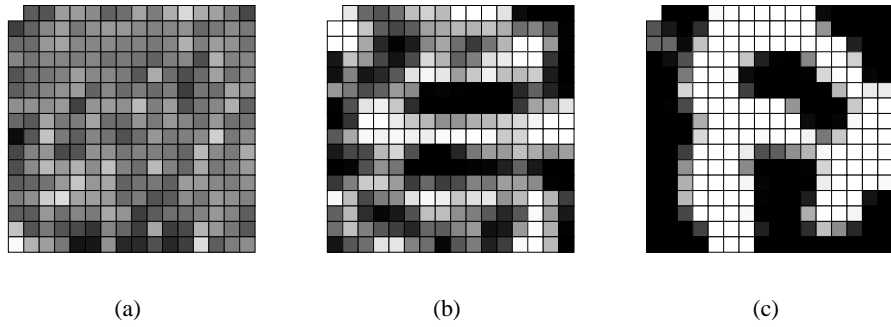

<div align="center">(a)                    (b)                    (c)</div>

Figure 2: Distribution of ODCs in the target cell sheet for different disparities between the bar stimuli driving the two afferent sheets. The gray level of each target cell indicates the relative strengths of projections from the two afferent sheets, where 'black' represents one and 'white' the other afferent sheet. (a) No disparity; (b) disparity: 50% of bar width (4 sensor pixels); (c) disparity: 75% of bar width (6 sensor pixels).

Several runs were performed for the three different disparities of the stimuli presented to the two sensors. Since the results for a given disparity were all qualitatively similar, we only show the results of one representative run for each value. The distribution of the formed ODCs in the target sheet is shown in Fig. 2, where the shading of each neuron indicates the relative numbers of projections from the two afferent sheets. In the absence of any disparity the formation of ODCs was suppressed. The residual ocular dominance modulations may be attributed to a small misalignment of the two chips with respect to the display. With the introduction of a disparity a very clear structure of ODCs emerges. The distribution of ODCs strongly depends on the disparity and does not vary significantly between runs for a given disparity. With increasing disparity the boundaries between ODCs become more distinct [9, 10]. The obtained maps are qualitatively similar to those obtained with simulated afferent inputs [1].

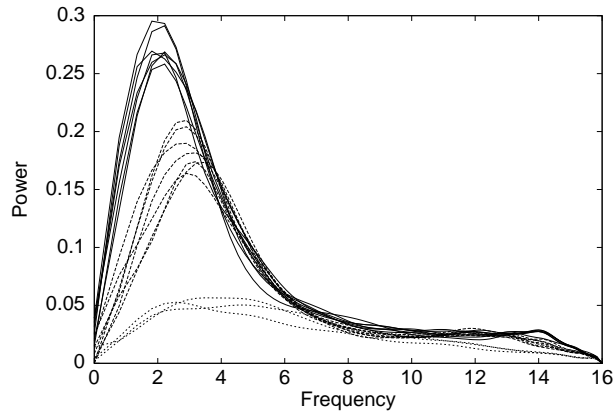

Figure 3: Power spectra of the spatial frequency distribution of ODCs in the target cell sheet for different disparities and data sets. A 'solid' line denotes data with disparity of 75% of bar width (6 sensor pixels); a 'dashed' line denotes a disparity of 50% of bar width (4 sensor pixels); a 'dotted' line denotes no disparity.

The power spectra obtained from two-dimensional Fourier transforms of the ODC distributions, represented in Fig. 3, show that the spatial frequency content of the ODCs is a function of disparity, consistent with experimental findings in the cat [8, 11, 12, 13], and that its variability between different runs of the same disparity is significantly smaller than between different disparities. The principal spatial frequency along each dimension of the target sheet is mainly determined by the NTF diffusion parameter [1] and the disparity. For the NTF diffusion parameter used here, it ranges between two and four cycles; increasing (decreasing) the diffusion parameter decreases (increases) the spatial frequency. The heights of the peaks show the degree of segregation, which increases with disparity, as already mentioned.

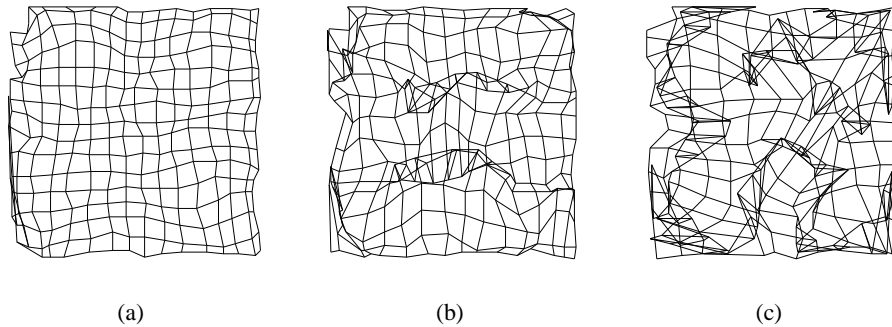

(a)                       (b)                       (c)

Figure 4: Topographic mapping between afferent sheets and target sheet for different disparities between the stimuli driving the two afferent sheets. The data are from the same runs as the ODC data of Fig. 2. (a) No disparity; (b) disparity: 50% of bar width (4 sensor pixels); (c) disparity: 75% of bar width (6 sensor pixels).

The resulting topographic maps for the same runs are shown in Fig. 4. In the absence of disparity the topographic map is almost perfect, with nearly one-to-one mapping between the afferent sheets and the target sheet, apart from remaining edge effects. However, disruptions appear at ODC boundaries in the runs with disparate stimuli, these disruptions becoming more distinct with increasing disparity due to the increasing sharpness of ODC boundaries.

The data presented above were obtained under suppression of spontaneous firing, so that each pixel generated exactly one spike in response to each moving bright-to-dark contrast boundary with an error rate of about 5%. By turning up the spontaneous firing rate we can test the robustness of the system to increased noise levels. We set the spontaneous firing rate to approximately 50%, so that roughly half of all spikes are not associated with an edge event. We also increased the bin size from 32 to 48 spikes per chip to compensate for the reduced intraocular correlations as a result of increased noise [3]. Fig. 5 shows a typical pattern of ODCs and the corresponding topographic map in the presence of 50% spontaneous activity. Although there are some distortions in the topographic map, in general it compares very favourably to maps developed in the absence of spontaneous activity. At an approximately 60% level of noise major disruptions in topographic map formation and attenuated ODC development are exhibited. Increasing the level of noise still further causes a complete breakdown of topographic and ODC map formation (data not shown).

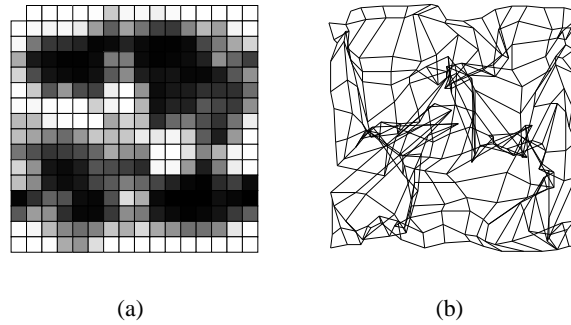

(a)                                        (b)

Figure 5: The pattern of ODCs and the topographic map that develop in the presence of approximately 50% noise. (a) The OD map; (b) the topographic map. The disparity is 50% of the bar width (4 sensor pixels).

## 7   Discussion

The refinement of topography and the development of ODCs can be robustly simulated with the considered hybrid system, consisting of an integrated analog visual sensing system that captures some of the key features of retinal processing and a mathematical model of activity-dependent synaptic competition. Despite the different structure of the input stimuli and the different noise characteristics of the real sensors from those used in the pure simulations [1], the results are comparable.

Several parameters of the vision sensors, such as refractory period and spontaneous firing rate, can be continuously varied with input bias voltages. This facilitates the evaluation of the performance of the model under different input conditions. The sensors were operated at long refractory periods, so that each pixel responded with a single spike to a contrast boundary moving across it. In this non-bursting mode the coding of the stimulus is very sparse, which makes the topographic refinement process more efficient [3].

The noise induced by the vision sensors manifests itself in occasionally missing responses of some pixels to a moving edge, in temporal jitter and a tunable level of spontaneous activity. With an optimal suppression of spontaneous firing, the error rate (number of missed and spurious events divided by total number of events) can be reduced to approximately 5%. Increased spontaneous activity levels show a strongly anisotropic distribution across the sensing arrays because of the inherent fixed-pattern noise present in the integrated sensors due to random mismatches in the fabricated circuits. This type of inhomogeneity has not been modeled in previous work. Spontaneous activity and mismatches between cells with the same functional role are prominent features of biological neural systems and biological information processing systems therefore have to deal with these nonidealities. The plasticity algorithm proves to be sufficiently robust with respect to these types of noise.

The developed ODC and topographic maps depend quite strongly on the disparity between the two sensors. At zero disparity, the formation of ODCs is practically suppressed and topography becomes very smooth. As the disparity increases, the period of the resulting ODCs increases, consistent with experimental results in the cat [8, 11, 12, 13], and, as expected, the degree of segregation also increases [9, 10]. In the presence of high levels of spontaneous activity in the afferent pathways, with as much as half of all spikes not being stimulus–related, the maps continue to exhibit well developed ODCs and topography. Although there are indications of distortions in the topographic maps in the presence of

approximately 50% spontaneous activity, the maps remain globally well structured. As spontaneous activity is increased further, map development becomes increasingly disrupted until it breaks down completely.

## 8    Conclusions

We examined the refinement of topographic mappings and the formation of ocular dominance columns by coupling a pair of integrated vision sensors to a neurotrophic model of synaptic plasticity. We have shown that the afferent input from real sensors looking at moving bar stimuli yields similar results as simulated partially randomized input and that these results are insensitive to the presence of significant noise levels.

### Acknowledgments

Tragically, J̈org Kramer died in July, 2002. TE dedicates this work to his memory.

TE thanks the Royal Society for the support of a University Research Fellowship. JK was supported in part by the Swiss National Foundation Research SPP grant. We thank David Lawrence of the Institute of Neuroinformatics for his invaluable help with interfacing the chip to the PC.

## References

[1] T. Elliott and N. R. Shadbolt, "A neurotrophic model of the development of the retinogeniculo-cortical pathway induced by spontaneous retinal waves," *Journal of Neuroscience*, vol. 19, pp. 7951–7970, 1999.

[2] A.K. McAllister, L.C. Katz, and D.C. Lo, "Neurotrophins and synaptic plasticity," *Annual Review of Neuroscience*, vol. 22, pp. 295–318, 1999.

[3] T. Elliott and J. Kramer, "Coupling an aVLSI neuromorphic vision chip to a neurotrophic model of synaptic plasticity: the development of topography," *Neural Computation*, vol. 14, pp. 2353–2370, 2002.

[4] J. Kramer, "An integrated optical transient sensor," *IEEE Trans. Circuits and Systems II: Analog and Digital Signal Processing*, 2002, submitted.

[5] J. Kramer, "An on/off transient imager with event-driven, asynchronous read-out," in *Proc. 2002 IEEE Int. Symp. on Circuits and Systems*, Phoenix, AZ, May 2002, vol. II, pp. 165–168, IEEE Press.

[6] T. Elliott and N. R. Shadbolt, "Multiplicative synaptic normalization and a nonlinear Hebb rule underlie a neurotrophic model of competitive synaptic plasticity," *Neural Computation*, vol. 14, pp. 1311–1322, 2002.

[7] T. Elliott and N. R. Shadbolt, "Competition for neurotrophic factors: Mathematical analysis," *Neural Computation*, vol. 10, pp. 1939–1981, 1998.

[8] G.J. Goodhill, "Topography and ocular dominance: a model exploring positive correlations," *Biological Cybernetics*, vol. 69, pp. 109–118, 1993.

[9] D.H. Hubel and T.N. Wiesel, "Binocular interaction in striate cortex of kittens reared with artificial squint," *Journal of Neurophysiology*, vol. 28, pp. 1041–1059, 1965.

[10] C.J. Shatz, S. Lindstr̈om, and T.N. Wiesel, "The distribution of afferents representing the right and left eyes in the cat's visual cortex," *Brain Research*, vol. 131, pp. 103–116, 1977.

[11] S. L̈owel, "Ocular dominance column development: Strabismus changes the spacing of adjacent columns in cat visual cortex," *Journal of Neuroscience*, vol. 14, pp. 7451–7468, 1994.

[12] G.J. Goodhill and S. L̈owel, "Theory meets experiment: correlated neural activity helps determine ocular dominance column periodicity," *Trends in Neurosciences*, vol. 18, pp. 437–439, 1995.

[13] S.B. Tieman and N. Tumosa, "Alternating monocular exposure increases the spacing of ocularity domains in area 17 of cats," *Visual Neuroscience*, vol. 14, pp. 929–938, 1997.
